# Hybrid Circuits of Interacting Computer Model and Biological Neurons

**Sylvie Renaud-LeMasson**[*]
Department of Physics
Brandeis University
Waltham, MA 02254

**Gwendal LeMasson**[#]
Department of Biology
Brandeis University
Waltham, MA 02254

**Eve Marder**
Department of Biology
Brandeis University
Waltham, MA 02254

**L.F. Abbott**
Department of Physics
Brandeis University
Waltham, MA 02254

## Abstract

We demonstrate the use of a digital signal processing board to construct hybrid networks consisting of computer model neurons connected to a biological neural network. This system operates in real time, and the synaptic connections are realistic effective conductances. Therefore, the synapses made from the computer model neuron are integrated correctly by the postsynaptic biological neuron. This method provides us with the ability to add additional, completely known elements to a biological network and study their effect on network activity. Moreover, by changing the parameters of the model neuron, it is possible to assess the role of individual conductances in the activity of the neuron, and in the network in which it participates.

[*]Present address, 1XL, Université de Bordeaux 1-Enserb, CNRSURA 846, 351 crs de la Liberation, 33405 Talence Cedex, France.

[#]Present address, LNPC, CNRS, Université de Bordeaux 1, Place de Dr. Peyneau, 33120 Arcachon, France

# 1 INTRODUCTION

A primary goal in neuroscience is to understand how the electrical properties of individual neurons contribute to the complex behavior of the networks in which they are found. However, the experimentalist wishing to assess the contribution of a given neuron or synapse in network function is hampered by lack of adequate tools. For example, although pharmacological agents are often used to block synaptic connections within a network (Eisen and Marder, 1982), or individual currents within a neuron (Tierney and Harris-Warrick, 1992), it is rarely possible to do precise pharmacological dissections of network function. Computational models of neurons and networks (Koch and Segev, 1989) allow the investigator the control over parameters not possible with pharmacology. However, because realistic computer models are always based on inadequate biophysical data, the investigator must always be concerned that the simulated system may differ from biological reality in a critical way. We have developed a system that allows us to construct hybrid networks consisting of a model neuron interacting with a biological neural network. This allows us to work with a real biological system while retaining complete control over the parameters of the model neuron.

# 2 THE MODEL NEURON

Biophysical data describing the ionic currents of the Lateral Pyloric (LP) neuron of the crab stomatogastric ganglion (STG) (Golowasch and Marder, 1992) were used to construct an isopotential model LP neuron using MAXIM. MAXIM is a software package that runs on MacIntosh systems and provides a graphical modeling tool for neurons and small neural networks (LeMasson, 1993). The model LP neuron used uses Hodgkin-Huxley type equations and contains a fast $Na^+$ conductance, a $Ca^+$ conductance, a delayed rectifier $K^+$ conductance, a transient outward current ($i_A$) and a hyperpolarization-activated current ($i_h$), as well as a leak conductance. This model is similar to that reported in Buchholtz et al. (1992) but because the raw data were refit using MAXIM, details are slightly different.

# 3 ARTIFICIAL SYNAPSES

Artificial chemical synapses are produced by the same method used in Sharp et al. (1993). An axoclamp in discontinuous current clamp (DCC) mode is used to record the membrane potential and inject current into the biological neurons (Fig. 1). The presynaptic membrane potential is used to control current injection into the postsynaptic neuron simulating a conductance change (rather than an injected current as in Yarom et al.). The synaptic current injected into the postsynaptic neuron depends on the programmed synaptic conductance and an investigator-determined reversal potential. The investigator also specifies the threshold and the function relating "transmitter release" to presynaptic membrane potential, as well as the time course of the synaptic conductance.

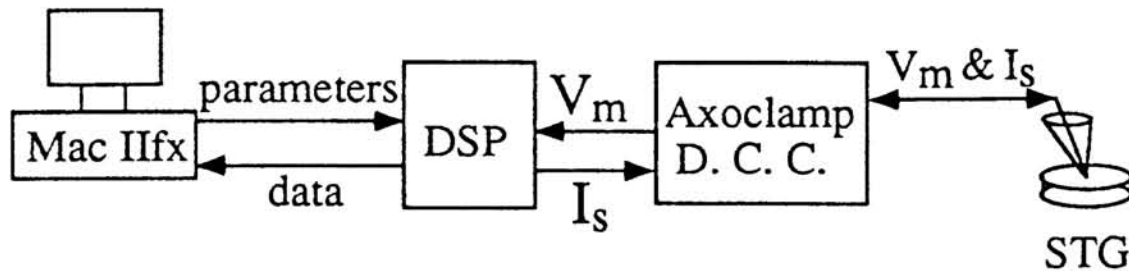

Figure 1: Schematic diagram of the system used to establish hybrid circuits.

## 4 HARDWARE

Our system uses a Digital Signal Processor (DSP) board with built-in A/D and D/A 16 bit-precision converters (Spectral Innovations MacDSP256KNI), with DSP32C (AT&T) mounted in a Macintosh II fx (MAC) computer. A block diagram of the system is shown Fig. 1. The parameters describing the membrane and synaptic conductances of the model neuron are stored in the MAC and are transferred to the DSP board RAM (256x32K) through the standard NuBus interface. The DSP translates the parameter files into look-up tables via a polynomial fitting procedure. The differential equations of the LP model and the artificial synapses are integrated by the DSP board, taking advantage of its optimized arithmetic functions and data access. In this system, the computational model runs on the DSP board, and the Mac IIfx functions to store and display data on the screen.

The computational speed of this system depends on the integration time step and the complexity of the model (the number of differential equations implemented in the model). For the results shown here, the integration time step was 0.7 msec, and under the conditions described below, 10-15 differential equations were used. The current system is limited to two real neurons and one model neuron because the DSP board has only two input and two output channels. A later generation system with more input and output channels and additional speed will increase the number of neurons and connections that can be created. During any one time step, the membrane potential of the model neuron is computed, the synaptic currents are determined, and a voltage command is exported to the Axoclamp instructing it to inject the appropriate current into the biological neuron (typically a few nA). During each time step the Axoclamp is used to measure the membrane potential of the biological neurons (typically between -80mV and 0mV) used to compute the value of the synaptic inputs to the model neuron. The computed and measured membrane potentials are periodically (every 500 time steps) sent to the computer main memory to be displayed and recorded.

To make this system run in real time, it is necessary to maintain perfect timing among all the components. Therefore in every experiment we first determine the minimum time step needed to do the integration depending on the complexity of the model being implemented. For complex models we used the internal clock of the MacII to drive the board. Under some conditions it was preferable to drive the board with an external clock. It is critical that the Axoclamp sampling rate be more than twice the board time step if the two are not synchronized. In our experiments, the Axoclamp switching

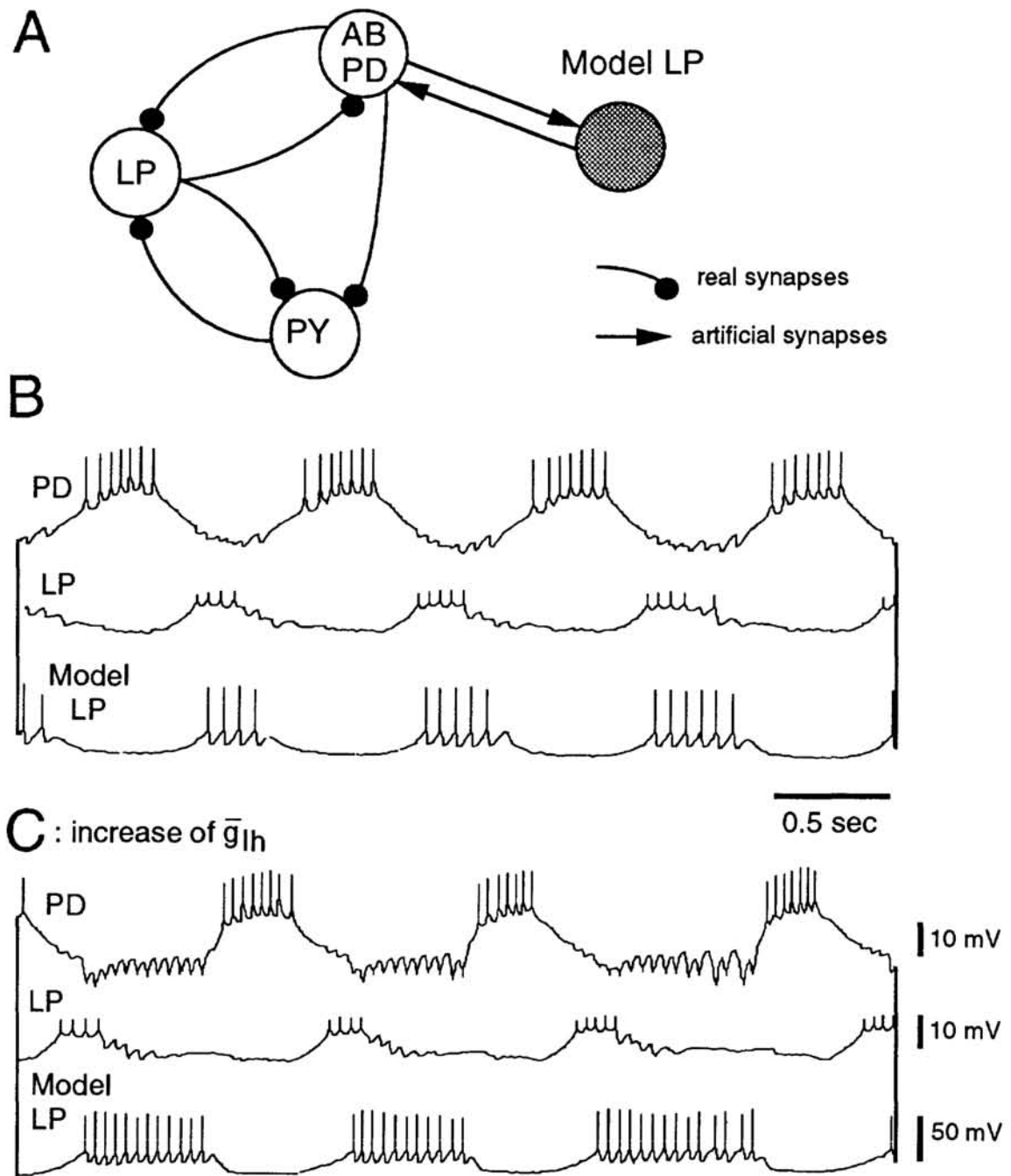

Figure 2: Hybrid network consisting of a model LP neuron connected to a PD neuron of a biological stomatogastric ganglion. A: Simplified connectivity diagram of the pyloric circuit of the stomatogastric ganglion. The AB/PD group consists of one AB neuron electrically coupled to two PD neurons. All chemical synapses are inhibitory. B: Simultaneous intracellular recordings from two biological neurons (PD and LP) and a plot of the membrane potential of the model LP neuron connected to the circuit. The parameters of the synaptic connections and the model LP neuron were adjusted so that the model LP neuron fired in the same time in the pyloric cycle as the biological LP neuron. C: Same recording configuration as B, but maximal conductance of $i_h$ in the model neuron was increased.

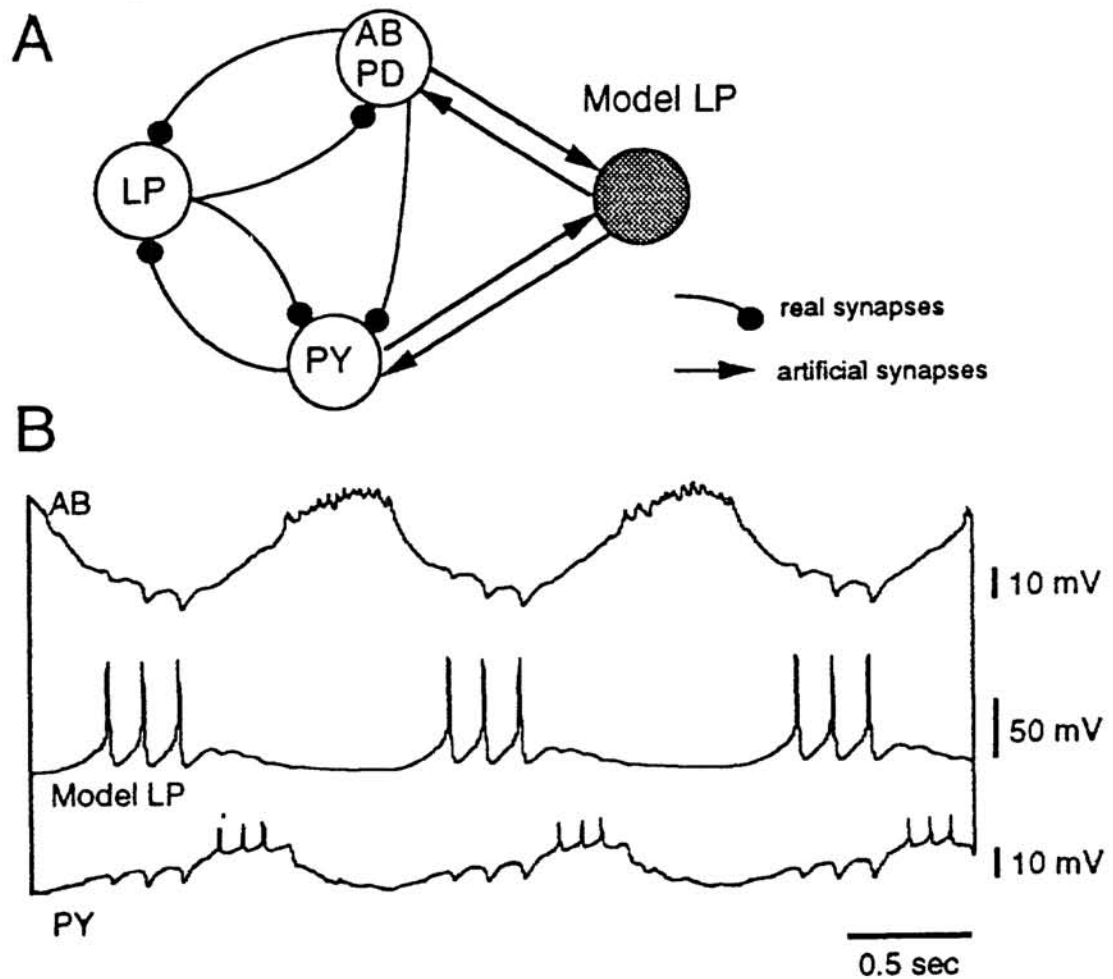

Figure 3: Hybrid network in which the model LP neuron is connected to two different biological neurons. A: Connectivity diagram showing the pattern of synaptic connections shown in part B. B: Simultaneous recordings from the biological AB neuron, the model LP neuron, and a biological PY neuron.

circuit was running about three times faster than the board time step. However, if experimental conditions force a slower Axoclamp sampling rate, then it will be important to synchronize the Axoclamp clock with the board.

## 5 RESULTS

The STG of the lobster, *Panulirus interruptus* contains one LP neuron, two Pyloric Dilator (PD) neurons, one Anterior Burster (AB), and eight Pyloric (PY) neurons (Eisen and Marder, 1982; Harris-Warrick et al., 1992). The connectivity among these neurons is known, and is shown in Figure 2A. The PD and LP neurons fire in alternation, because of the reciprocal inhibitory connections between them. Figure 2B shows a model LP neuron connected with reciprocal inhibitory synapses to a biological PD neuron. The parameters controlling the threshold, activation curve, time course, and reversal potential of the model neuron were adjusted until the model neuron fired at the same time within the rhythmic pyloric cycle as the biological LP neuron (Fig. 2B). Once these parameters were set, it was then possible to ask what effect changing the membrane properties of the model neuron had on emergent network activity. Figure 2C shows the result of increasing the maximal conductance of one of the currents in the model LP neuron, $i_h$. Note that increasing this current

increased the number of LP action potentials per burst. The increased activity in the LP neuron delayed the onset of the next burst in the PD neurons because of the inhibitory synapse between the model LP neuron and the biological PD neuron, and the cycle period therefore also increased. Another effect seen in this example, is that the increased conductance of $i_h$ in the LP neuron delayed the onset of the model LP neuron's firing relative to that of the biological LP neuron.

In the experiment shown in Figure 3 we created reciprocal inhibitory connections between the model LP neuron and two biological neurons, the AB and a PY (Fig. 3A). (The action potentials in the AB neuron are highly attenuated by the cable properties of this neuron). This example shows clearly the unitary inhibitory postsynaptic potentials (IPSPs) in the biological neurons resulting from the model LP's action potentials. During each burst of LP action potentials the IPSPs in the AB neuron increase considerably in amplitude, although the AB neuron's membrane potential is moving towards the reversal potential of the IPSPs. This occurs presumably because the conductance of the AB neuron is higher right at the end of its burst, and decreases as it hyperpolarizes. The same burst of LP action potentials evokes IPSPs in the PY neuron that increase in amplitude, here presumably because the PY neuron is depolarizing and increasing the driving force on the artificial chemical synapse. These recordings demonstrate that although the same function is controlling the synaptic "release" properties in the model LP neuron, the actual change in membrane potential evoked by action potentials in the LP neuron is affected by the total conductance of the biological neurons.

# 6 CONCLUSIONS

The ability to connect a realistic model neuron to a biological network offers a unique opportunity to study the effects of individual currents on network activity. It also provides realistic, two-way interactions between biological and computer-based networks. As well as providing an important new tool for neuroscience, this represents an exciting new direction in biologically-based computing.

# 7 ACKNOWLEDGMENTS

We thank Ms. Joan McCarthy for help with manuscript preparation. Research supported by MH 46742, the Human Science Frontier Program, and NSF DMS-9208206.

# 8 REFERENCES

Buchholtz, F., Golowasch, J., Epstein, I.R., and Marder, E. (1992) Mathematical model of an identified stomatogastric ganglion neuron. *J. Neurophysiology* 67:332-340.

Eisen, J.S., and Marder, E. (1982) Mechanisms underlying pattern generation in lobster stomatogastric ganglion as determined by selective inactivation of identified neurons. III. Synaptic connections of electrically coupled pyloric neurons. *J. Neurophysiology* 48:1392-1415.

Golowasch, J. and Marder, E. (1992) Ionic currents of the lateral pyloric neuron of

the stomatogastric ganglion of the crab. *J. Neurophysiology* 67:318-331.

Harris-Warrick, R.M., Marder, E., Selverston, A.I., and Maurice, M., eds. (1992) *Dynamic Biological Networks*.  Cambridge, MA: MIT Press.

Koch, C., and Segev, I., eds. (1989) *Methods in Neuronal Modeling*.  Cambridge, MA: MIT press.

LeMasson, G. (1993) Maxim: A software system for simulating single neurons and neural networks, in preparation.

Sharp, A.A., O'Neil, M.B., Abbott, L.F. and Marder, E. (1993) The dynamic clamp: Computer-generated conductances in real neurons. *J. Neurophysiology*, in press.

Yarom, Y. (1992) Rhythmogenesis in a hybrid system interconnecting an olivary neuron to an network of coupled oscillators. *Neuroscience* 44:263-275.